# Analog VLSI Implementation of Multi-dimensional Gradient Descent

**David B. Kirk, Douglas Kerns, Kurt Fleischer, Alan H. Barr**
California Institute of Technology
Beckman Institute 350-74
Pasadena, CA 91125
E-mail: dk@egg.gg.caltech.edu

## Abstract

We describe an analog VLSI implementation of a multi-dimensional gradient estimation and descent technique for minimizing an on-chip scalar function $f()$. The implementation uses noise injection and multiplicative correlation to estimate derivatives, as in [Anderson, Kerns 92]. One intended application of this technique is setting circuit parameters on-chip automatically, rather than manually [Kirk 91]. Gradient descent optimization may be used to adjust synapse weights for a backpropagation or other on-chip learning implementation. The approach combines the features of continuous multi-dimensional gradient descent and the potential for an annealing style of optimization. We present data measured from our analog VLSI implementation.

## 1  Introduction

This work is similar to [Anderson, Kerns 92], but represents two advances. First, we describe the extension of the technique to multiple dimensions. Second, we demonstrate an implementation of the multi-dimensional technique in analog VLSI, and provide results measured from the chip. Unlike previous work using noise sources in adaptive systems, we use the noise as a means of estimating the gradient of a function $f(y)$, rather than performing an annealing process [Alspector 88]. We also estimate gradients continuously in position and time, in contrast to [Umminger 89] and [Jabri 91], which utilize discrete position gradient estimates.

It is interesting to note the existence of related algorithms, also presented in this volume [Cauwenberghs 93] [Alspector 93] [Flower 93]. The main difference is that our implementation operates in continuous time, with continuous differentiation and integration operators. The other approaches realize the integration and differentiation processes as discrete addition and subtraction operations, and use unit perturbations. [Cauwenberghs 93] provides a detailed derivation of the convergence and scaling properties of the discrete approach, and a simulation. [Alspector 93] provides a description of the use of the technique as part of a neural network hardware architecture, and provides a simulation. [Flower 93] derived a similar discrete algorithm from a node perturbation perspective in the context of multi-layered feedforward networks. Our work is similar in spirit to [Dembo 90] in that we don't make any explicit assumptions about the "model" that is embodied in the function $f()$. The function may be implemented as a neural network. In that case, the gradient descent is on-chip learning of the parameters of the network.

We have fabricated a working chip containing the continuous-time multi-dimensional gradient descent circuits. This paper includes chip data for individual circuit components, as well as the entire circuit performing multi-dimensional gradient descent and annealing.

## 2   The Gradient Estimation Technique

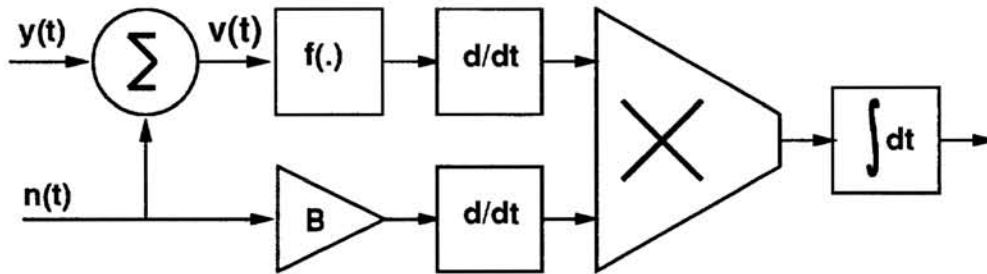

Figure 1: Gradient estimation technique from [Anderson, Kerns 92]

Anderson and Kerns [Anderson, Kerns 92] describe techniques for one-dimensional gradient estimation in analog hardware. The gradient is estimated by correlating (using a multiplier) the output of a scalar function $f(v(t))$ with a noise source $n(t)$, as shown in Fig. 1. The function input $y(t)$ is additively "contaminated" by the noise $n(t)$ to produce $v(t) = y(t) + n(t)$. A scale factor $B$ is used to set the scale of the noise to match the function output, which improves the signal-to-noise ratio. The signals are "high-pass" filtered to approximate differentiation (shown as $d/dt$ operators in Fig. 1) directly before the multiplication. The results of the multiplication are "low-pass" filtered to approximate integration.

The gradient estimate is integrated over time, to smooth out some of the noise and to damp the response. This smoothed estimate is compared with a "zero" reference, using an amplifier $A$, and the result is fed back to the input, as shown in Fig. 2. The contents of Fig. 1 are represented by the "Gradient Estimation" box in Fig. 2.

We have chosen to implement the multi-dimensional technique in analog VLSI. We

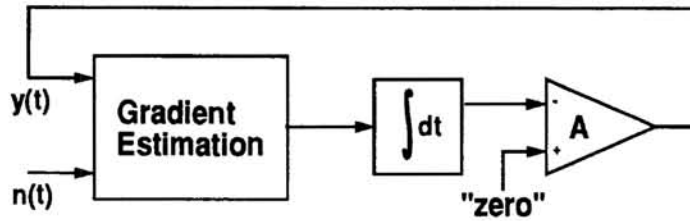

Figure 2: Closing the loop: performing gradient descent using the gradient estimate.

will not reproduce here the one-dimensional analysis from [Anderson, Kerns 92], but summarize some of the more important results, and provide a multi-dimensional derivation. [Anderson 92] provides a more detailed theoretical discussion.

## 3   Multi-dimensional Derivation

The multi-dimensional gradient descent operation that we are approximating can be written as follows:

$$\underline{y}'(t) = -k\nabla f(\underline{y}(t)) \tag{1}$$

where $\underline{y}$ and $\underline{y}'$ are vectors, and the solution is obtained continuously in time $t$, rather than at discrete $t_i$. The circuit described in the block diagram in Fig. 1 computes an approximation to the gradient:

$$\nabla f = \frac{\partial f}{\partial y_i} \approx \int \left( \frac{d}{dt} f\left(\underline{y}(t) + \underline{n}(t)\right) n_i'(t) \right) dt \tag{2}$$

We approximate the operations of differentiation and integration in time by realizable high-pass and low-pass filters, respectively. To see that Eq. 2 is valid, and that this result is useful for approximating Eq. 1, we sketch an $N$-dimensional extension of [Anderson 92]. Using the chain rule,

$$\frac{d}{dt} f\left(\underline{y}(t) + \underline{n}(t)\right) = \sum_j \left(y_j'(t) + n_j'(t)\right) \frac{\partial f}{\partial y_j} \tag{3}$$

Assuming $n_j'(t) \gg y_j'(t)$, the rhs is approximated to produce

$$\frac{d}{dt} f\left(\underline{y}(t) + \underline{n}(t)\right) = \sum_j n_j'(t) \frac{\partial f}{\partial y_j} \tag{4}$$

Multiplying both sides by $n_i'(t)$, and taking the expectation integral operator $E[\ ]$ of each side,

$$E\left[ n_i'(t) \frac{d}{dt} f\left(\underline{y}(t) + \underline{n}(t)\right) \right] = E\left[ n_i'(t) \sum_j n_j'(t) \frac{\partial f}{\partial y_j} \right] \tag{5}$$

If the noise sources $n_i(t)$ and $n_j(t)$ are uncorrelated, $n_i'(t)$ is independent of $n_j'(t)$ when $i \neq j$, and the sum on the right has a contribution only when $i = j$,

$$E\left[ n_i'(t) \frac{d}{dt} f\left(\underline{y}(t) + \underline{n}(t)\right) \right] = E\left[ n_i'(t) n_i'(t) \frac{\partial f}{\partial y_i} \right] \tag{6}$$

$$E\left[n_i'(t)\frac{d}{dt}f\left(\underline{y}(t)+\underline{n}(t)\right)\right] \approx \alpha\frac{\partial f}{\partial y_i} = \alpha\nabla f \tag{7}$$

The expectation operator $E[\,]$ can be used to smooth random variations of the noise $n_i(t)$. So, we have

$$\nabla f = \frac{\partial f}{\partial y_i} \approx \frac{1}{\alpha}E\left[n_i'(t)\frac{d}{dt}f\left(\underline{y}(t)+\underline{n}(t)\right)\right] \tag{8}$$

Since the descent rate $k$ is arbitrary, we can absorb $\alpha$ into $k$. Using equation 8, we can approximate the gradient descent technique as follows:

$$y_i'(t) \approx -\hat{k}\ E\left[n_i'(t)\frac{d}{dt}f\left(\underline{y}(t)+\underline{n}(t)\right)\right] \tag{9}$$

## 4    Elements of the Multi-dimensional Implementation

We have designed, fabricated, and tested a chip which allows us to test these ideas. The chip implementation can be decomposed into six distinct parts:

**noise source(s):** an analog VLSI circuit which produces a noise function. An independent, correlation-free noise source is needed for each input dimension, designated $n_i(t)$. The noise circuit is described in [Alspector 91].

**target function:** a scalar function $f(y_1, y_2, \cdots, y_N)$ of $N$ input variables, bounded below, which is to be minimized [Kirk 91]. The circuit in this case is a 4-dimensional variant of the bump circuit described in [Delbrück 91]. In the general case, this $f()$ can be any scalar function or error metric, computed by some circuit. Specifically, the function may be a neural network.

**input signal(s):** the inputs $y_i(t)$ to the function $f()$. These will typically be on-chip values, or real-world inputs.

**multiplier circuit(s):** the multiplier computes the correlation between the noise values and the function output. Offsets in the multiplication appear as systematic errors in the gradient estimate, so it is important to compensate for the offsets. Linearity is not especially important, although monotonicity is critical. Ideally, the multiplication will also have a "tanh-like" character, limiting the output range for extreme inputs.

**integrator:** an integration over time is approximated by a low-pass filter

**differentiator:** the time derivatives of the noise signals and the function are approximated by a high-pass filter.

The $N$ inputs, $y_i(t)$, are additively "contaminated" with the noise signals, $n_i(t)$, by capacitive coupling, producing $v_i(t) = y_i(t) + n_i(t)$, the inputs to the function $f()$. The function output is differentiated, as are the noise functions. Each differentiated noise signal is correlated with the differentiated function output, using the multipliers. The results are low-pass filtered, providing $N$ partial derivative estimates, for the $N$ input dimensions, shown for 4 dimensions in Fig. 3.

The function $f()$ is implemented as an 4-dimensional extension of Delbrück's [Delbrück 91] bump circuit. Details of the $N$-dimensional bump circuit can be found in [Kirk 93]. For learning and other applications, the function $f()$ can implement some other error metric to be minimized.

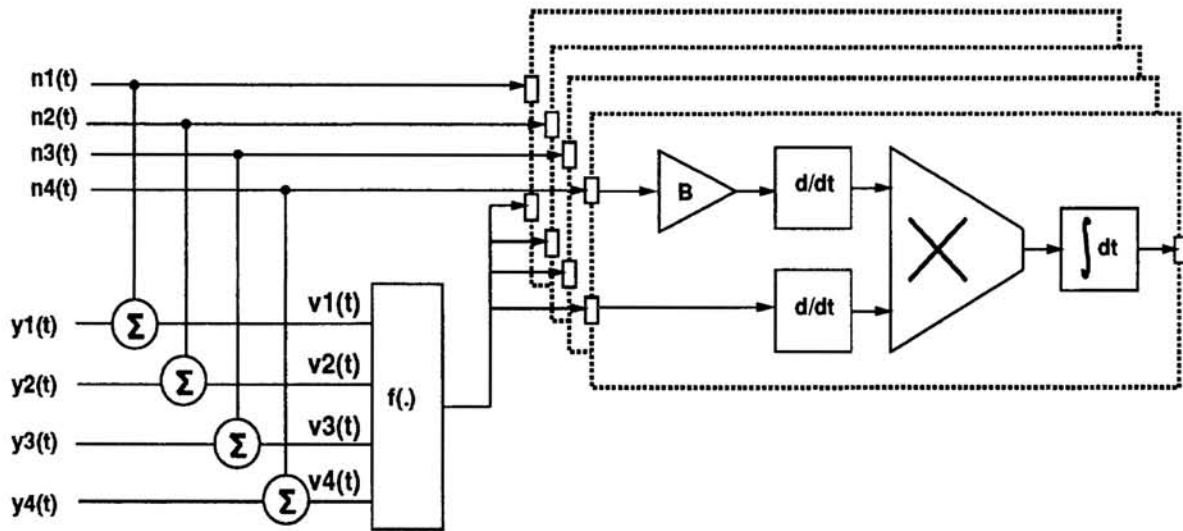

Figure 3: Block diagram for a 4-dimensional gradient estimation circuit.

## 5   Chip Results

We have tested chips implementing the gradient estimation and gradient descent techniques described in this paper. Figure 4 shows the gradient estimate, without the closed loop descent process. Figure 5 shows the trajectories of two state variables during the 2D gradient descent process. Figure 6 shows the gradient descent process in operation on a 2D bump surface, and Fig. 7 shows how, using appropriate choice of noise scale, we can perform annealing using the gradient estimation hardware.

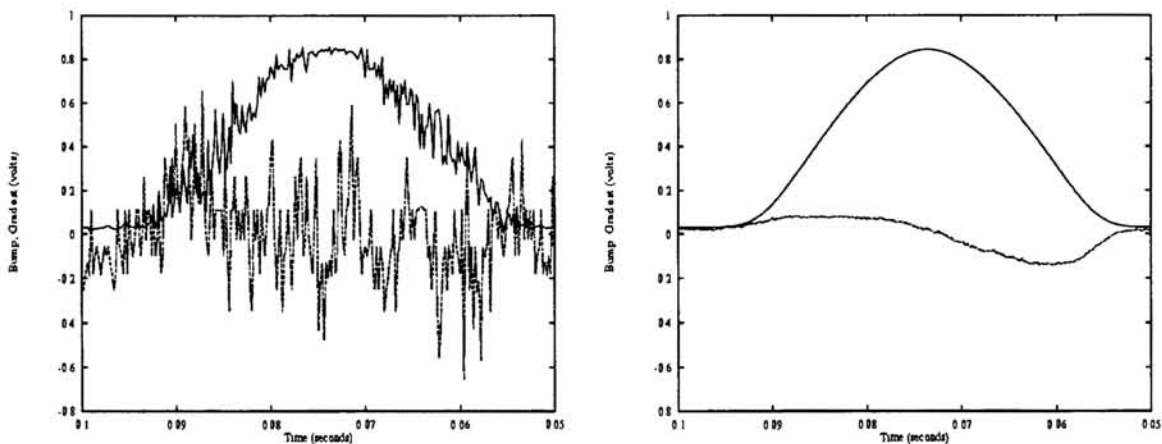

Figure 4: Measured Chip Data: 1D Gradient Estimate. Upper curves are 1D bump output as the input $y(t)$ is a slow triangle wave. Lower curves are gradient estimates. (left) raw data, and (right) average of 1024 runs.

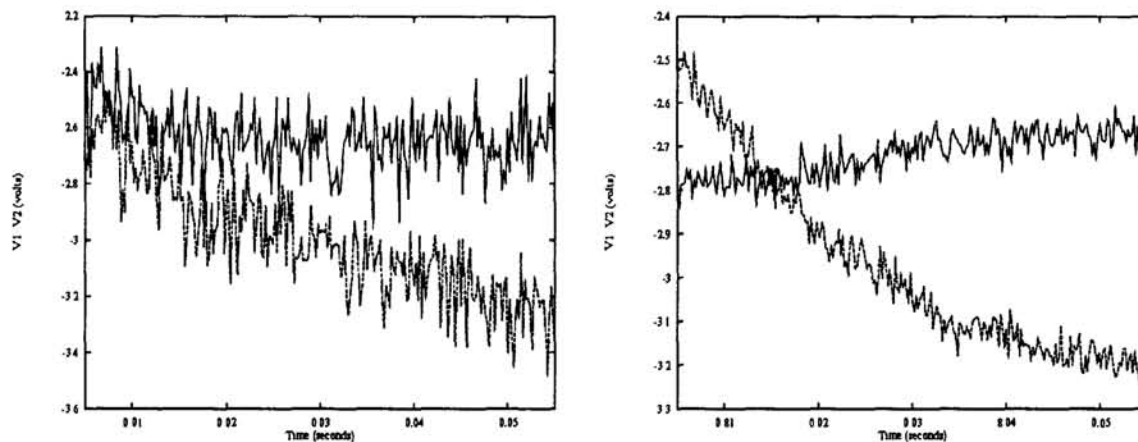

Figure 5: Measured Chip Data: 2D Gradient Descent. The curves above show the function optimization by gradient descent for 2 variables. Each curve represents the path of one of the state variables $\underline{y}(t)$ from some initial values to the values for which the function $f()$ is minimized. (left) raw data, and (right) average of 8 runs.

## 6    Conclusions

We have implemented an analog VLSI structure for performing continuous multi-dimensional gradient descent, and the gradient estimation uses only local information. The circuitry is compact and easily extensible to higher dimensions. This implementation leads to on-chip multi-dimensional optimization, such as is needed to perform on-chip learning for a hardware neural network. We can also perform a kind of annealing by adding a schedule to the scale of the noise input. Our approach also has some drawbacks, however. The gradient estimation is sensitive to the input offsets in the multipliers and integrators, since those offsets result in systematic errors. Also, the gradient estimation technique adds noise to the input signals.

We hope that with only small additional circuit complexity, the performance of analog VLSI circuits can be greatly increased by permitting them to be intrinsically adaptive. On-chip implementation of an approximate gradient descent technique is an important step in this direction.

### Acknowledgements

This work was supported in part by an AT&T Bell Laboratories Ph.D. Fellowship, and by grants from Apple, DEC, Hewlett Packard, and IBM. Additional support was provided by NSF (ASC-89-20219), as part of the NSF/DARPA STC for Computer Graphics and Scientific Visualization. All opinions, findings, conclusions, or recommendations expressed in this document are those of the author and do not necessarily reflect the views of the sponsoring agencies.

## References

[Alspector 93] Alspector, J., R. Meir, B. Yuhas, and A. Jayakumar, "A Parallel Gradient

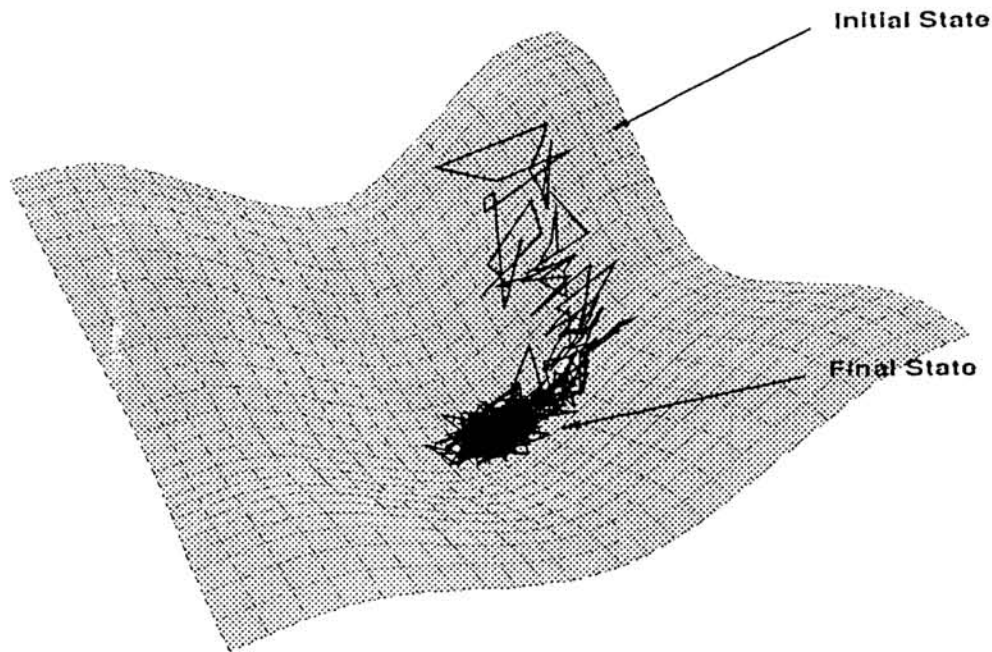

Figure 6: Measured Chip Data: 2D Gradient Descent. Here we see the results for 2D gradient descent on a 2D bump surface. Both the bump surface and the descent path are actual data measured from our chips.

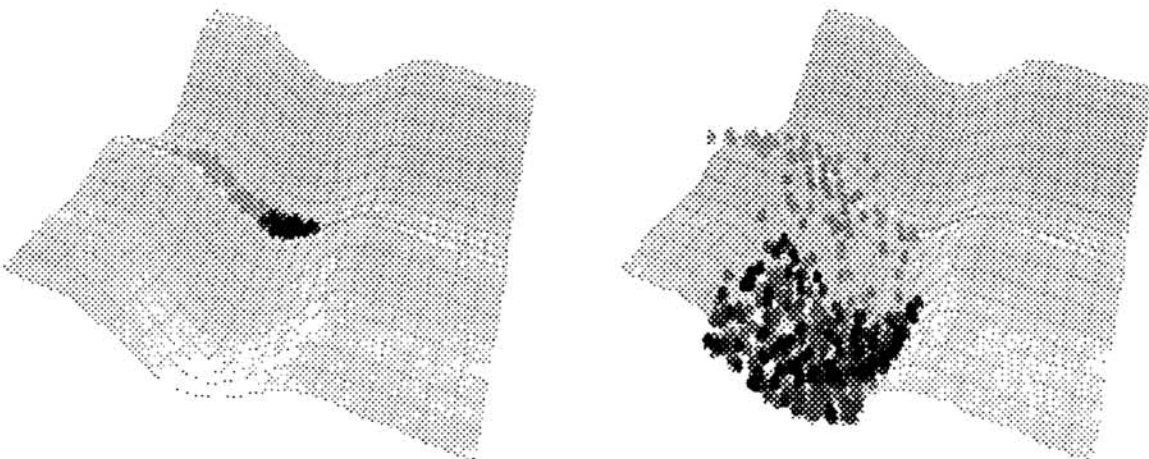

Figure 7: Measured Chip Data: 2D Gradient Descent and Annealing. Here we see the effects of varying the amplitude of the noise. The dots represent points along the optimization path. At left, with small magnitude noise, the process descends to a local minimum. At right, with larger magnitude, the descent process escapes to the global minimum. A schedule of gradually decreasing noise amplitude could reduce the probability of getting caught in undesirable local minima, and increase the probability of converging to a small region near a more desirable minimum, or even the global minimum.

Descent Method for Learning in Analog VLSI Neural Networks," in *Advances in Neural Information Processing Systems*, Vol. 5, Morgan Kaufman, San Mateo, CA, 1993.

[Alspector 91] Alspector, J., J. W. Gannett, S. Haber, M. B. Parker, and R. Chu, "A VLSI–Efficient Technique for Generating Multiple Uncorrelated Noise Sources and Its Application to Stochastic Neural Networks," IEEE Transactions on Circuits and Systems, Vol.38, no.1, pp.109–123, January, 1991.

[Alspector 88] Alspector, J., B. Gupta, and R. B. Allen, "Performance of a stochastic learning microchip," in *Advances in Neural Information Processing Systems, vol. I*, Denver Colorado, Nov. 1988. D. S. Touretzky, ed., Morgan Kauffman Publishers, 1989, pp. 748–760.

[Anderson, Kerns 92] Anderson, Brooke P., and Douglas Kerns, "Using Noise Injection and Correlation in Analog Hardware to Estimate Gradients," submitted to IEEE Transactions on Circuits and Systems I: Fundamental Theory and Applications.

[Anderson 92] Anderson, Brooke P., "Low-pass Filters as Expectation Operators for Multiplicative Noise," submitted to IEEE Transactions on Circuits and Systems I: Fundamental Theory and Applications.

[Cauwenberghs 93] Cauwenberghs, Gert, "A Fast Stochastic Error-Descent Algorithm for Supervised Learning and Optimization," in *Advances in Neural Information Processing Systems*, Vol. 5, Morgan Kaufman, San Mateo, CA, 1993.

[Delbrück 91] Delbrück, Tobias, " 'Bump' Circuits for Computing Similarity and Dissimilarity of Analog Voltages," Proceedings of International Joint Conference on Neural Networks, July 8-12, 1991, Seattle Washington, pp I-475–479. (Extended version as Caltech Computation and Neural Systems Memo Number 10.)

[Dembo 90] Dembo, A., and T. Kailath, "Model-Free Distributed Learning," IEEE Transactions on Neural Networks, Vol. 1, No. 1, pp. 58-70, 1990.

[Flower 93] Flower, B., and M. Jabri, "Summed Weight Neuron Perturbation: An $\mathcal{O}(n)$ Improvement over Weight Perturbation," in *Advances in Neural Information Processing Systems*, Vol. 5, Morgan Kaufman, San Mateo, CA, 1993.

[Jabri 91] Jabri, M., S. Pickard, P. Leong, Z. Chi, and B. Flower, "Architectures and Implementations of Right Ventricular Apex Signal Classifiers for Pacemakers," IEEE Neural Information Processing Systems 1991 (NIPS 91), Morgan Kaufman, San Diego, 1991.

[Kerns 92] Kerns, Douglas, "A Compact Noise Source for VLSI Applications," submitted to IEEE Transactions on Circuits and Systems I: Fundamental Theory and Applications.

[Kirk 91] Kirk, David, Kurt Fleischer, and Alan Barr, "Constrained Optimization Applied to the Parameter Setting Problem for Analog Circuits," IEEE Neural Information Processing Systems 1991 (NIPS 91), Morgan Kaufman, San Diego, 1991.

[Kirk 93] Kirk, David, "Accurate and Precise Computation using Analog VLSI, with Applications to Computer Graphics and Neural Networks," Ph.D. Thesis, California Institute of Technology, Caltech-CS-TR-93-??, June, 1993.

[Mead 89] Mead, Carver, "Analog VLSI and Neural Systems," Addison-Wesley, 1989.

[Platt 89] Platt, John, "Constrained Optimization for Neural Networks and Computer Graphics," Ph.D. Thesis, California Institute of Technology, Caltech-CS-TR-89-07, June, 1989.

[Umminger 89] Umminger, Christopher B., and Steven P. DeWeerth, "Implementing Gradient Following in Analog VLSI," Advanced Research in VLSI, MIT Press, Boston, 1989, pp. 195-208.